# Autoencoders, Minimum Description Length and Helmholtz Free Energy

**Geoffrey E. Hinton**
Department of Computer Science
University of Toronto
6 King's College Road
Toronto M5S 1A4, Canada

**Richard S. Zemel**
Computational Neuroscience Laboratory
The Salk Institute
10010 North Torrey Pines Road
La Jolla, CA 92037

## Abstract

An autoencoder network uses a set of *recognition* weights to convert an input vector into a code vector. It then uses a set of *generative* weights to convert the code vector into an approximate reconstruction of the input vector. We derive an objective function for training autoencoders based on the Minimum Description Length (MDL) principle. The aim is to minimize the information required to describe both the code vector and the reconstruction error. We show that this information is minimized by choosing code vectors stochastically according to a Boltzmann distribution, where the generative weights define the energy of each possible code vector given the input vector. Unfortunately, if the code vectors use distributed representations, it is exponentially expensive to compute this Boltzmann distribution because it involves all possible code vectors. We show that the recognition weights of an autoencoder can be used to compute an approximation to the Boltzmann distribution and that this approximation gives an upper bound on the description length. Even when this bound is poor, it can be used as a Lyapunov function for learning both the generative and the recognition weights. We demonstrate that this approach can be used to learn factorial codes.

## 1   INTRODUCTION

Many of the unsupervised learning algorithms that have been suggested for neural networks can be seen as variations on two basic methods: Principal Components Analysis (PCA)

and Vector Quantization (VQ) which is also called clustering or competitive learning. Both of these algorithms can be implemented simply within the autoencoder framework (Baldi and Hornik, 1989; Hinton, 1989) which suggests that this framework may also include other algorithms that combine aspects of both. VQ is powerful because it uses a very non-linear mapping from the input vector to the code but weak because the code is a purely local representation. Conversely, PCA is weak because the mapping is linear but powerful because the code is a distributed, factorial representation. We describe a new objective function for training autoencoders that allows them to discover non-linear, factorial representations.

## 2    THE MINIMUM DESCRIPTION LENGTH APPROACH

One method of deriving a cost function for the activities of the hidden units in an autoencoder is to apply the Minimum Description Length (MDL) principle (Rissanen 1989). We imagine a communication game in which a *sender* observes an ensemble of training vectors and must then communicate these vectors to a *receiver*. For our purposes, the sender can wait until all of the input vectors have been observed before communicating any of them – an *online* method is not required. Assuming that the components of the vectors are finely quantized we can ask how many bits must be communicated to allow the receiver to reconstruct the input vectors perfectly. Perhaps the simplest method of communicating the vectors would be to send each component of each vector separately. Even this simple method requires some further specification before we can count the number of bits required. To send the value, $x_{i,c}$, of component $i$ of input vector $c$ we must encode this value as a bit string. If the sender and the receiver have already agreed on a probability distribution that assigns a probability $p(x)$ to each possible quantized value, $x$, Shannon's coding theorem implies that $x$ can be communicated at a cost that is bounded below by $-\log p(x)$ bits. Moreover, by using block coding techniques we can get arbitrarily close to this bound so we shall treat it as the true cost. For coding real values to within a quantization width of $t$ it is often convenient to assume a Gaussian probability distribution with mean zero and standard deviation $\sigma$. Provided that $\sigma$ is large compared with $t$, the cost of coding the value $x$ is then $-\log t + 0.5 \log 2\pi\sigma^2 + x^2/2\sigma^2$.

This simple method of communicating the training vectors is generally very wasteful. If the components of a vector are correlated it is generally more efficient to convert the input vector into some other representation before communicating it. The essence of the MDL principle is that the best model of the data is the one that minimizes the total number of bits required to communicate it, including the bits required to describe the coding scheme. For an autoencoder it is convenient to divide the total description length into three terms. An input vector is communicated to the receiver by sending the activities of the hidden units and the residual differences between the true input vector and the one that can be reconstructed from the hidden activities. There is a *code cost* for the hidden activities and a *reconstruction cost* for the residual differences. In addition there is a one-time *model cost* for communicating the weights that are required to convert the hidden activities into the output of the net. This model cost is generally very important within the MDL framework, but in this paper we will ignore it. In effect, we are considering the limit in which there is so much data that this limited model cost is negligible.

PCA can be viewed as a special case of MDL in which we ignore the model cost and we limit the code cost by only using $m$ hidden units. The question of how many bits are required

to code each hidden unit activity is also ignored. Thus the only remaining term is the reconstruction cost. Assuming that the residual differences are encoded using a zero-mean Gaussian with the same predetermined variance for each component, the reconstruction cost is minimized by minimizing the squared differences.

Similarly, VQ is a version of MDL in which we limit the code cost to at most $\log m$ bits by using only $m$ winner-take-all hidden units, we ignore the model cost, and we minimize the reconstruction cost.

In standard VQ we assume that each input vector is converted into a specific code. Surprisingly, it is more efficient to choose the codes stochastically so that the very same input vector is sometimes communicated using one code and sometimes using another. This type of "stochastic VQ" is exactly equivalent to maximizing the log probability of the data under a mixture of Gaussians model. Each code of the VQ then corresponds to the mean of a Gaussian and the probability of picking the code is the posterior probability of the input vector under that Gaussian. Since this derivation of the mixture of Gaussians model is crucial to the new techniques described later, we shall describe it in some detail.

## 2.1   The "bits-back" argument

The description length of an input vector using a particular code is the sum of the code cost and reconstruction cost. We define this to be the *energy* of the code, for reasons that will become clear later. Given an input vector, we define the *energy* of a code to be the sum of the code cost and the reconstruction cost. If the prior probability of code $i$ is $\pi_i$ and its squared reconstruction error is $d_i^2$ the energy of the code is

$$E_i = -\log \pi_i - k \log t + \frac{k}{2} \log 2\pi\sigma^2 + \frac{d^2}{2\sigma^2} \tag{1}$$

where $k$ is the dimensionality of the input vector, $\sigma^2$ is the variance of the fixed Gaussian used for encoding the reconstruction errors and $t$ is the quantization width.

Now consider the following situation: We have fitted a VQ to some training data and, for a particular input vector, two of the codes are equally good in the sense that they have equal energies. In a standard VQ we would gain no advantage from the fact that there are two equally good codes. However, the fact that we have a choice of two codes should be worth something. It does not matter which code we use so if we are vague about the choice of code we should be able to save one bit when communicating the code.

To make this argument precise consider the following communication game: The sender is already communicating a large number of random bits to the receiver, and we want to compute the *additional* cost of communicating some input vectors. For each input vector we have a number of alternative codes $h_1...h_i...h_m$ and each code has an energy, $E_i$. In a standard VQ we would pick the code, $j$, with the lowest energy. But suppose we pick code $i$ with a probability $p_i$ that depends on $E_i$. Our expected cost then appears to be higher since we sometimes pick codes that do not have the minimum value of $E$.

$$< \text{Cost} >= \sum_i p_i E_i \tag{2}$$

where $< ... >$ is used to denote an expected value. However, the sender can use her freedom of choice in stochastically picking codes to communicate some of the random

bits that need to be communicated anyway. It is easy to see how random bits can be used to stochastically choose a code, but it is less obvious how these bits can be recovered by the receiver, because he is only sent the chosen code and does not know the probability distribution from which it was picked. This distribution depends on the particular input vector that is being communicated. To recover the random bits, the receiver waits until all of the training vectors have been communicated losslessly and then runs exactly the same learning algorithm as the sender used. This allows the receiver to recover the recognition weights that are used to convert input vectors into codes, even though the only weights that are explicitly communicated from the sender to the receiver are the generative weights that convert codes into approximate reconstructions of the input. After learning the recognition weights, the receiver can reconstruct the probability distribution from which each code was stochastically picked because the input vector has already been communicated. Since he also knows which code was chosen, he can figure out the random bits that were used to do the picking. The expected number of random bits required to pick a code stochastically is simply the entropy of the probability distribution over codes

$$H = -\sum_i p_i \log p_i \tag{3}$$

So, allowing for the fact that these random bits have been successfully communicated, the true expected combined cost is

$$F = \sum_i p_i E_i - H \tag{4}$$

Note that $F$ has exactly the form of Helmholtz free energy. It can be shown that the probability distribution which minimizes F is

$$p_i = \frac{e^{-E_i}}{\sum_j e^{-E_j}} \tag{5}$$

This is exactly the posterior probability distribution obtained when fitting a mixture of Gaussians to an input vector.

The idea that a stochastic choice of codes is more efficient than just choosing the code with the smallest value of E is an example of the concept of stochastic complexity (Rissanen, 1989) and can also be derived in other ways. The concept of stochastic complexity is unnecessarily complicated if we are only interested in fitting a mixture of Gaussians. Instead of thinking in terms of a stochastically chosen code plus a reconstruction error, we can simply use Shannon's coding theorem directly by assuming that we code the input vectors using the mixture of Gaussians probability distribution. However, when we start using more complicated coding schemes in which the input is reconstructed from the activities of several different hidden units, the formulation in terms of F is much easier to work with because it liberates us from the constraint that the probability distribution over codes must be the optimal one. There is generally no efficient way of computing the optimal distribution, but it is nevertheless possible to use F with a suboptimal distribution as a Lyapunov function for learning (Neal and Hinton, 1993). In MDL terms we are simply using a suboptimal coding scheme in order to make the computation tractable.

One particular class of suboptimal distributions is very attractive for computational reasons. In a factorial distribution the probability distribution over $m^d$ alternatives factors into $d$ independent distributions over $m$ alternatives. Because they can be represented compactly,

factorial distributions can be computed conveniently by a non-stochastic feed-forward recognition network.

# 3   FACTORIAL STOCHASTIC VECTOR QUANTIZATION

Instead of coding the input vector by a single, stochastically chosen hidden unit, we could use several different pools of hidden units and stochastically pick one unit in each pool. All of the selected units within this distributed representation are then used to reconstruct the input. This amounts to using several different VQs which cooperate to reconstruct the input. Each VQ can be viewed as a dimension and the chosen unit within the VQ is the value on that dimension. The number of possible distributed codes is $m^d$ where $d$ is the number of VQs and $m$ is the number of units within a VQ. The weights from the hidden units to the output units determine what output is produced by each possible distributed code. Once these weights are fixed, they determine the reconstruction error that would be caused by using a particular distributed code. If the prior probabilities of each code are also fixed, Eq. 5 defines the optimal probability distribution over distributed codes, where the index $i$ now ranges over the $m^d$ possible codes.

Computing the correct distribution requires an amount of work that is exponential in $d$, so we restrict ourselves to the suboptimal distributions that can be factored into $d$ independent distributions, one for each VQ. The fact that the correct distribution is not really factorial will not lead to major problems as it does in mean field approximations of Boltzmann machines (Galland, 1993). It will simply lead to an overestimate of the description length but this overestimate can still be used as a bound when learning the weights. Also the excess bits caused by the non-independence will force the generative weights towards values that cause the correct distribution to be approximately factorial.

## 3.1   Computing the Expected Reconstruction Error

To perform gradient descent in the description length given in Eq. 4, it is necessary to compute, for each training example, the derivative of the expected reconstruction cost with respect to the activation probability of each hidden unit. An obvious way to approximate this derivative is to use Monte Carlo simulations in which we stochastically pick one hidden unit in each pool. This way of computing derivatives is faithful to the underlying stochastic model, but it is inevitably either slow or inaccurate. Fortunately, it can be replaced by a fast exact method when the output units are linear and there is a squared error measure for the reconstruction. Given the probability, $h_i^v$, of picking hidden unit $i$ in VQ $v$, we can compute the expected reconstructed output $y_j$ for output unit $j$ on a given training case

$$y_j = b_j + \sum_v w_{ji}^v h_i^v \tag{6}$$

where $b_j$ is the bias of unit $j$ and $w_{ji}^v$ is the generative weight from $i$ to $j$ in VQ $v$. We can also compute the variance in the reconstructed output caused by the stochastic choices within the VQs. Under the assumption that the stochastic choices within different VQs are independent, the variances contributed by the different VQs can simply be added.

$$V_j = \sum_v \sum_i h_i^v \left( w_{ji}^v - \sum_k w_{jk}^v h_k^v \right)^2 \tag{7}$$

The expected squared reconstruction error for each output unit is $V_j + (y_j - d_j)^2$ where $d_j$ is the desired output. So if the reconstruction error is coded assuming a zero-mean Gaussian distribution the expected reconstruction cost can be computed exactly[1]. It is therefore straightforward to compute the derivatives, with respect to any weight in the network, of all the terms in Eq. 4.

## 4   AN EXAMPLE OF FACTORIAL VECTOR QUANTIZATION

Zemel (1993) presents several different data sets for which factorial vector quantization (FVQ) produces efficient encodings. We briefly describe one of those examples. The data set consists of 200 images of simple curves as shown in figure 1. A network containing 4 VQs, each containing 6 hidden units, is trained on this data set. After training, the final outgoing weights for the hidden units are as shown in figure 2. Each VQ has learned to represent the height of the spline segment that connects a pair of control points. By chaining these four segments together the image can be reconstructed fairly accurately. For new images generated in the same way, the description length is approximately 18 bits for the reconstruction cost and 7 bits for the code. By contrast, a stochastic vector quantizer with 24 hidden units in a single competing group has a reconstruction cost of 36 bits and a code cost of 4 bits. A set of 4 separate stochastic VQs each of which is trained on a different $8x3$ vertical slice of the image also does slightly worse than the factorial VQ (by 5 bits) because it cannot smoothly blend the separate segments of the curve together. A purely linear network with 24 hidden units that performs a version of principal components analysis has a slightly lower reconstruction cost but a much higher code cost.

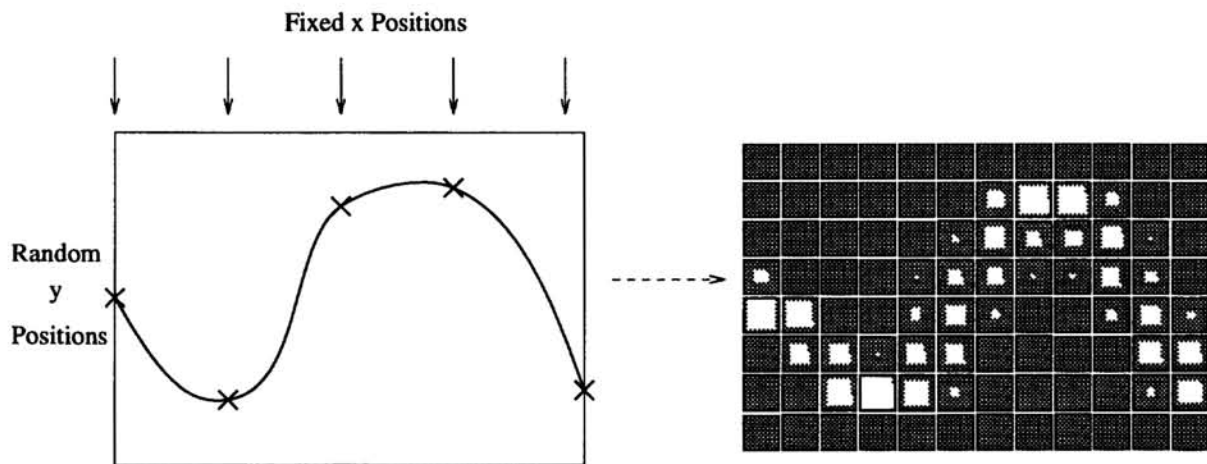

Figure 1: Each image in the spline dataset is generated by fitting a spline to 5 control points with randomly chosen $y$-positions. An image is formed by blurring the spline with a Gaussian. The intensity of each pixel is indicated by the area of white in the display. The resulting images are 8x12 pixels, but have only 5 underlying degrees of freedom.

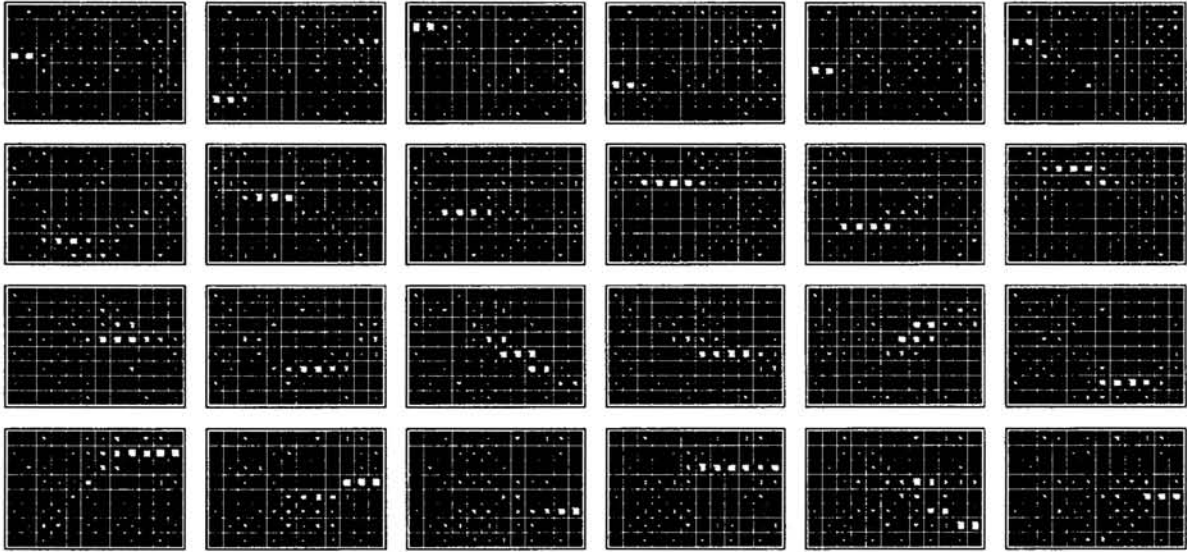

Figure 2: The outgoing weights of the hidden units for a network containing 4 VQs with 6 units in each, trained on the spline dataset. Each 8x12 weight block corresponds to a single unit, and each row of these blocks corresponds to one VQ.

## 5  DISCUSSION

A natural approach to unsupervised learning is to use a generative model that defines a probability distribution over observable vectors. The obvious maximum likelihood learning procedure is then to adjust the parameters of the model so as to maximize the sum of the log probabilities of a set of observed vectors. This approach works very well for generative models, such as a mixture of Gaussians, in which it is tractable to compute the expectations that are required for the application of the EM algorithm. It can also be applied to the wider class of models in which it is tractable to compute the derivatives of the log probability of the data with respect to each model parameter. However, for non-linear models that use distributed codes it is usually intractable to compute these derivatives since they require that we integrate over all of the exponentially many codes that could have been used to generate each particular observed vector.

The MDL principle suggest a way of making learning tractable in these more complicated generative models. The optimal way to code an observed vector is to use the correct posterior probability distribution over codes given the current model parameters. However, we are free to use a suboptimal probability distribution that is easier to compute. The description length using this suboptimal method can still be used as a Lyapunov function for learning the model parameters because it is an upper bound on the optimal description length. The excess description length caused by using the wrong distribution has the form of a Kullback-Liebler distance and acts as a penalty term that encourages the recognition weights to approximate the correct distribution as well as possible.

There is an interesting relationship to statistical physics. Given an input vector, each possible code acts like an alternative configuration of a physical system. The function

$E$ defined in Eq. 1 is the energy of this configuration. The function $F$ in Eq. 4 is the Helmholtz free energy which is minimized by the thermal equilibrium or Boltzmann distribution. The probability assigned to each code at this minimum is exactly its posterior probability given the parameters of the generative model. The difficulty of performing maximum likelihood learning corresponds to the difficulty of computing properties of the equilibrium distribution. Learning is much more tractable if we use the *non-equilibrium* Helmholtz free energy as a Lyapunov function (Neal and Hinton, 1993). We can then use the recognition weights of an autoencoder to compute some non-equilibrium distribution. The derivatives of $F$ encourage the recognition weights to approximate the equilibrium distribution as well as they can, but we do not need to reach the equilibrium distribution before adjusting the generative weights that define the energy function of the analogous physical system.

In this paper we have shown that an autoencoder network can learn factorial codes by using non-equilibrium Helmholtz free energy as an objective function. In related work (Zemel and Hinton 1994) we apply the same approach to learning population codes. We anticipate that the general approach described here will be useful for a wide variety of complicated generative models. It may even be relevant for gradient descent learning in situations where the model is so complicated that it is seldom feasible to consider more than one or two of the innumerable ways in which the model could generate each observation.

## Acknowledgements

This research was supported by grants from the Ontario Information Technology Research Center, the Institute for Robotics and Intelligent Systems, and NSERC. Geoffrey Hinton is the Noranda Fellow of the Canadian Institute for Advanced Research. We thank Peter Dayan, Yann Le Cun, Radford Neal and Chris Williams for helpful discussions.

## Footnotes

[1]Each VQ contributes non-Gaussian noise and the combined noise is also non-Gaussian. But since its variance is known, the expected cost of coding the reconstruction error using a Gaussian prior can be computed exactly. The fact that this prior is not ideal simply means that the computed reconstruction cost is an upper bound on the cost using a better prior.

## References

Baldi, P. and Hornik, K. (1989) Neural networks and principal components analysis: Learning from examples without local minima. *Neural Networks*, **2**, 53-58.

Galland, C. C. (1993) The limitations of deterministic Boltzmann machine learning. *Network*, **4**, 355-379.

Hinton, G. E. (1989) Connectionist learning procedures. *Artificial Intelligence*, **40**, 185-234.

Neal, R., and Hinton, G. E. (1993) A new view of the EM algorithm that justifies incremental and other variants. *Manuscript available from the authors.*

Rissanen, J. ( 1989) *Stochastic Complexity in Statistical Inquiry.* World Scientific Publishing Co., Singapore.

Zemel, R. S. (1993) *A Minimum Description Length Framework for Unsupervised Learning.* PhD. Thesis, Department of Computer Science, University of Toronto.

Zemel, R. S. and Hinton, G. E. (1994) Developing Population Codes by Minimizing Description Length. In J. Cowan, G. Tesauro, and J. Alspector (Eds.), *Advances in Neural Information Processing Systems 6*, San Mateo, CA: Morgan Kaufmann.